# Hierarchical Memory-Based Reinforcement Learning

**Natalia Hernandez-Gardiol**
Artificial Intelligence Lab
Massachusetts Institute of Technology
Cambridge, MA 02139
*nhg@ai.mit.edu*

**Sridhar Mahadevan**
Department of Computer Science
Michigan State University
East Lansing, MI 48824
*mahadeva@cse.msu.edu*

## Abstract

A key challenge for reinforcement learning is scaling up to large partially observable domains. In this paper, we show how a hierarchy of behaviors can be used to create and select among variable length short-term memories appropriate for a task. At higher levels in the hierarchy, the agent abstracts over lower-level details and looks back over a variable number of high-level decisions in time. We formalize this idea in a framework called Hierarchical Suffix Memory (HSM). HSM uses a memory-based SMDP learning method to rapidly propagate delayed reward across long decision sequences. We describe a detailed experimental study comparing memory vs. hierarchy using the HSM framework on a realistic corridor navigation task.

## 1 Introduction

Reinforcement learning encompasses a class of machine learning problems in which an agent learns from experience as it interacts with its environment. One fundamental challenge faced by reinforcement learning agents in real-world problems is that the state space can be very large, and consequently there may be a long delay before reward is received. Previous work has addressed this issue by breaking down a large task into a hierarchy of subtasks or abstract behaviors [1, 3, 5].

Another difficult issue is the problem of perceptual aliasing: different real-world states can often generate the same observations. One strategy to deal with perceptual aliasing is to add memory about past percepts. Short-term memory consisting of a linear (or tree-based) sequence of primitive actions and observations has been shown to be a useful strategy [2]. However, considering short-term memory at a flat, uniform resolution of primitive actions would likely scale poorly to tasks with long decision sequences. Thus, just as spatio-temporal abstraction of the state space improves scaling in completely observable environments, for large partially observable environments a similar benefit may result if we consider the space of *past experience* at variable resolution. Given a task, we want a hierarchical strategy for rapidly bringing to bear past experience that is appropriate to the grain-size of the decisions being considered.

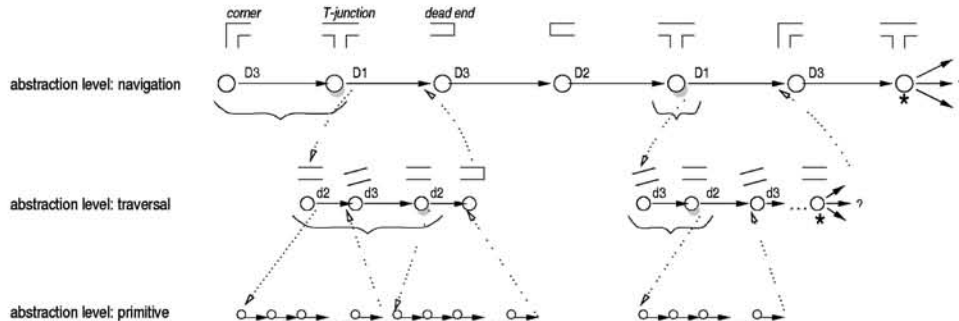

Figure 1: This figure illustrates memory-based decision making at two levels in the hierarchy of a navigation task. At each level, each decision point (shown with a star) examines its past experience to find states with similar history (shown with shadows). At the abstract (navigation) level, observations and decisions occur at intersections. At the lower (corridor-traversal) level, observations and decisions occur within the corridor.

In this paper, we show that considering past experience at a variable, task-appropriate resolution can speed up learning and greatly improve performance under perceptual aliasing. The resulting approach, which we call Hierarchical Suffix Memory (HSM), is a general technique for solving large, perceptually aliased tasks.

## 2 Hierarchical Suffix Memory

By employing short-term memory over abstract decisions, each of which involves a hierarchy of behaviors, we can apply memory at a *more informative* level of abstraction. An important side-effect is that the agent can look at a decision point many steps back in time while ignoring the exact sequence of low-level observations and actions that transpired. Figure 1 illustrates the HSM framework.

The problem of learning under perceptual aliasing can be viewed as discovering an informative sequence of past actions and observations (that is, a history suffix) for a given world state that enables an agent to act optimally in the world. We can think of each situation in which an agent must choose an action (a choice point) as being labeled with a pair $[\sigma, l]$: $l$ refers to the abstraction level and $\sigma$ refers to the history suffix. In the completely observable case, $\sigma$ has a length of one, and decisions are made based on the current observation. In the partially observable case, we must additionally consider past history when making decisions. In this case, the suffix $\sigma$, is some sequence of past observations and actions that must be learned. This idea of representing memory as a variable-length suffix derives from work on learning approximations of probabilistic suffix automata [2, 4].

Here is the general HSM procedure (including model-free and model-based updates):

1. Given an abstraction level $l$ and choice point $s$ within $l$: for each potential future decision, $d$, examine the history at level $l$ to find a set of past choice points that have executed $d$ and whose incoming (suffix) history most closely matches that of the current point. Call this set of instances the "voting set" for decision $d$.

2. Choose $d_t$ as the decision with the highest average discounted sum of reward over the voting set. Occasionally, choose $d_t$ using an exploration strategy.

Here, $t$ is the event counter of the current choice point at level $l$.

3. Execute the decision $d_t$ and record: $o_t$, the resulting observation; $r_t$, the reward received; and $n_t$, the duration of abstract action $d_t$ (measured by the number of primitive environment transitions executed by the abstract action).

   Note that for every environment transition from state $s_{i-1}$ to state $s_i$ with reward $r_i$ and discount $\gamma$, we accumulate any reward and update the discount factor: $\qquad r_t \leftarrow r_t + \gamma_t r_i \qquad \gamma_t \leftarrow \gamma \gamma_t$

4. Update the Q-value for the current decision point and for each instance in the voting set using the decision, reward, and duration values recorded along with the instance.

   Model-free: use an SMDP Q-learning update rule ($\beta$ is the learning rate):

$$Q_l(s_t, d_t) \leftarrow (1 - \beta)Q_l(s_t, d_t) + \beta(r_t + \gamma_t \max_d Q_l(s_{t+n_t}, d))$$

   Model-based: if a state-transition model is being used, a sweep of value iteration can be executed[1]. Let the state corresponding to the decision point at time $t$ be represented by the suffix $s$:

$$Q_l(s, d_t) \leftarrow R_l(s, d_t) + \sum_{s'} P_l(s' \mid s, d_t)V_l(s')(\gamma^{N_{d_t}})$$

   where $R_l(s, d_t)$ is the estimated immediate reward from executing decision $d_t$ from the choice point $[s, l]$; $P_l(s' \mid s, d_t)$ is the estimated probability that the agent arrives in $[s', l]$ given that it executed $d_t$ from $[s, l]$; $V_l(s')$ is the utility of the situation $[s', l]$; and $N_{d_t}$ is the average duration of the transition $[s, l]$ to $[s', l]$ under abstract action $d_t$.

HSM requires a technique for short-term memory. We implemented the Nearest Sequence Memory (NSM) and Utile Suffix Memory (USM) algorithms proposed by McCallum [2]. NSM records each of its raw experiences as a linear chain. To choose the next action, the agent evaluates the outcomes of the $k$ "nearest" neighbors in the experience chain. NSM evaluates the closeness between two states according to the match length of the suffix chain preceding the states. The chain can either be grown indefinitely, or old experiences can be replaced after the chain reaches a maximum length. With NSM, a model-free learning method, HSM uses an SMDP Q-learning rule as described above. USM also records experience in a linear time chain. However, instead of attempting to choose actions based on a greedy history match, USM tries to explicitly determine how much memory is useful for predicting reward. To do this, the agent builds a tree-like structure for state representation online, selectively adding depth to the tree if the additional history distinction helps to predict reward. With USM, which learns a model, HSM updates the Q-values by doing one sweep of value iteration with the leaves of the tree as states.

Finally, to implement the hierarchy of behaviors, in principle any hierarchical reinforcement learning method may be used. For our implementation, we used the Hierarchy of Abstract Machines (HAM) framework proposed by Parr and Russell [3]. When executed, an abstract machine executes a partial policy and returns control to the caller upon termination. The HAM architecture uses a Q-learning rule modified for SMDPs.

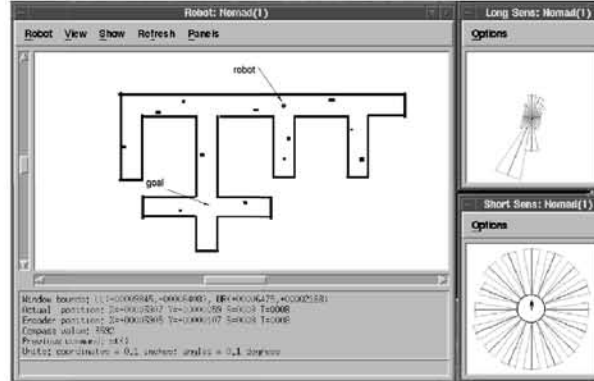

Figure 2: The corridor environment in the Nomad 200 robot simulator. The goal is the 4-way junction. The robot is shown at the middle T-junction. The robot is equipped with 16 short-range infrared and long-range sonar sensors. The other figures in the environment are obstacles around which the robot must maneuver.

## 3   The Navigation Task

To test the HSM framework, we devised a navigation task in a simulated corridor environment (see Figure 2). The task is for the robot to find its way from the start, the center T-junction, to the goal, the four-way junction. The robot receives a reward at the goal intersection and and a small negative reward for each primitive step taken.

Our primary testbed was a simulated agent using a Nomad 200 robot simulator. This simulated robot is equipped with 20 bumper and 16 sonar and infrared sensors, arranged radially. The dynamics of the simulator are not "grid world" dynamics: the Nomad 200 simulator represents continuous, noisy sensor input and the occasional unreliability of actuators. The environment presents significant perceptual ambiguity. Additionally, sensor readings can be noisy; even if the agent is at the goal or an intersection, it might not "see" it. Note the size of the robot relative to the environment in Figure 2.

What makes the task difficult are the several activities that must be executed concurrently. Conceptually, there are two levels to our navigation problem. At the top, most abstract, level is the root task of navigating to the goal. At the lower level is the task of physically traversing the corridors, avoiding obstacles, maintaining alignment with the walls, etc.

## 4   Implementation of the Learning Agents

In our experiments, we compared several learning agents: a basic HAM agent, four agents using HSM (each using a different short-term memory technique), and a "flat" NSM agent.

To build a set of behaviors for hallway navigation, we used a three-level hierarchy. The top abstract level is basically a choice state for choosing a hallway navigation direction (see Figure 3a). In each of the four nominal directions (front, back, left, right), the agent can make one of three observations: wall, open, or unknown. The agent must learn to choose among the four abstract machines to reach the next

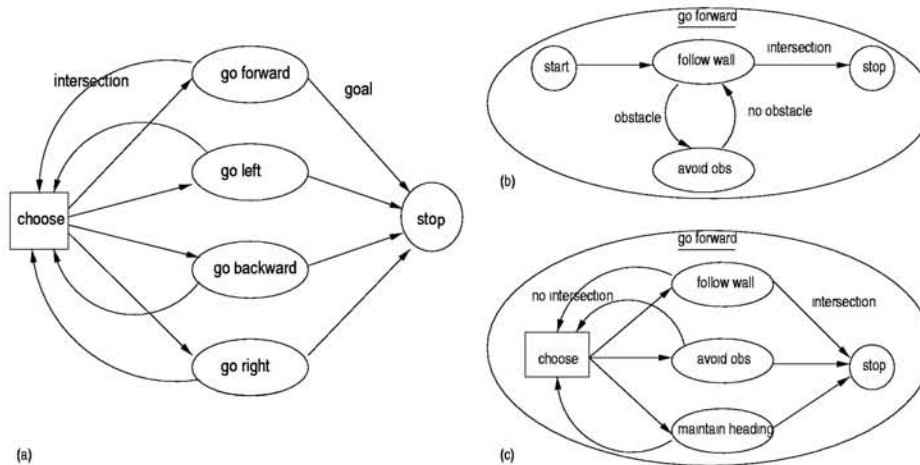

Figure 3: Hierarchical structure of behaviors for hallway navigation. Figure (a) shows the most abstract level – responsible for navigating in the environment. Figures (b) and (c) show two implementations of the hall-traversal machines. The machine in Figure (b) is reactive, and Figure (c) is a machine with a choice point.

intersection. This top level machine has control initially, and it regains control at intersections. The second level of the hierarchy contains the machines for traversing the hallway. The traversal behavior is shown in Figure 3b. Each of the four machines at this level executes a reactive strategy for traversing a corridor. Finally, the third level of the hierarchy implements the follow-wall and avoid-obstacle strategies using primitive actions. Both the avoid-obstacle and the follow-wall strategies were themselves trained previously using Q-learning to exploit the power of reuse in the hierarchical framework.

The HAM agent uses a three-level behavior hierarchy as described above. There is a single choice state, at the top level, and the agent learns to coordinate its choices by keeping a table of Q-values. The Q-value table is indexed by the current percepts and the chosen action (one of four abstract machines). The HAM agent uses a discount of 0.9, and a learning rate of 0.1. Exploration is done with a simple epsilon-greedy strategy.

The first pair of HSM agents use the same behavior hierarchy as the HAM agent. However, they use short-term memory at the most abstract level to learn a strategy for navigating the corridor. The first of these agents uses NSM at the top level with a history length of 1000, $k = 4$, a discount of 0.9, and a learning rate of 0.1. The second agent uses USM at the top level with a discount of 0.95. The performance of these top-level memory agents was studied as a control against the more complex multi-level memory agents described next.

The next pair of HSM agents use short-term memory *both* at the abstract navigation level and at the intermediate level. The behavior decomposition at the abstract navigation level is the same for the previous agents; however, the traversal behavior is in turn composed of machines that must make a decision based on short-term memory. Each of the machines at the traversal level uses short-term memory to learn to coordinate a strategy behaviors for traversing a corridor. The memory-based version of the traversal machine is shown in Figure 3c. The first of these agents uses NSM as the short-term memory technique at both levels of the hierarchy.

It uses a history length of 1000, $k = 4$, a discount of 0.9, and a learning rate of 0.1. The second agent uses USM as the short-term memory technique at the top level with a discount of 0.95. At the intermediate level, it uses NSM with the same learning parameters as the preceding agent. Exploration is done with a simple epsilon-greedy strategy in all cases.

Finally, we study the behavior of a "flat" NSM agent. The flat agent must keep track of the following perceptual data: first, it needs the same perceptual information as the top-level HAM (so it can identify the goal); second, it needs the additional perceptual data for aligning to walls and for avoiding obstacles: whether it was bumped, and the angle to the wall (binned into 4 groups of 45° each). The flat agent chooses among four primitive actions: go-forward, veer-left, veer-right, and back-up. Not only must it learn to make it to the goal, it must simultaneously learn to align itself to walls and avoid obstacles. The NSM agent uses a history length of 1000 , $k = 4$, a discount of 0.9, and a learning rate of 0.1. Exploration is done with a simple epsilon-greedy strategy.

## 5  Experimental Results

In Figure 4, we see the learning performance of each agent in the navigation task. The graphs show the performance advantage of both multi-level HSM agents over the other agents. In particular, we find that the flat memory-based agent does considerably worse than the other three, as expected. The flat agent must carry around the perceptual data to perform both high and low-level behaviors. From the point of view of navigation, this results in long strings of uninformative corridor states between the more informative intersection states. Since takes such an agent longer to discover patterns in its experience, it never quite learns to navigate successfully to the goal.

Next, both multi-level memory-based hierarchical agents outperform the HAM agent. The HAM agent does better at navigation than the flat agent since it abstracts away the perceptually aliased corridor states. However, it is unable to distinguish between all of the intersections. Without the ability to tell which T-junctions lead to the goal, and which to a dead end, the HAM agent does not perform as well. The multi-level HSM agents also outperform the single-level ones. The multi-level agents can tune their traversing strategy to the characteristics of the cluttered hallway by using short-term memory at the intermediate level.

Finally, although it initially does worse, the multi-level HSM agent with USM soon outperforms the multi-level HSM agent with NSM. This is because the USM algorithm forces the agent to learn a state representation that uses only as much incoming history as needed to predict reward. That is, it tries to learn the right history suffix for each situation rather approximating the suffix by simply matching greedily on incoming history. Learning such a representation takes some time, but, once learned, produces better performance.

## 6  Conclusions and Future Work

In this paper we described a framework for solving large perceptually aliased tasks called Hierarchical Suffix Memory (HSM). This approach uses a hierarchical behavioral structure to index into past memory at multiple levels of resolution. Organizing past experience hierarchically scales better to problems with long decision sequences. We presented an experiment comparing six different learning methods, showing that hierarchical short-term memory produces overall the best performance

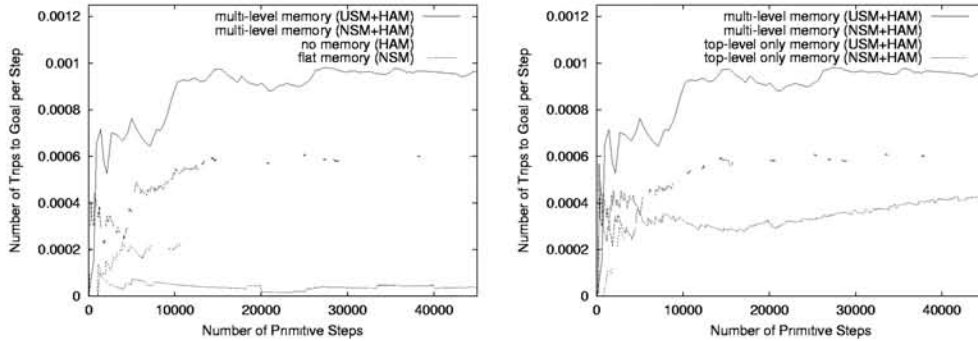

Figure 4: Learning performance in the navigation task. Each curve is averaged over eight trials for each agent.

in a perceptually aliased corridor navigation task.

One key limitation of the current HSM framework is that each abstraction level examines only the history at its own level. Allowing interaction between the memory streams at each level of the hierarchy would be beneficial. Consider a navigation task in which the decision at a given intersection depends on an observation seen while traversing the corridor. In this case, the abstract level should have the ability to "zoom in" to inspect a particular low-level experience in greater detail. We expect that pursuit of general frameworks such as HSM to manage past experience at variable granularity will lead to strategies for control that are able to gracefully scale to large, partially observable problems.

## Acknowledgements

This research was carried out while the first author was at the Department of Computer Science and Engineering, Michigan State University. This research is supported in part by a KDI grant from the National Science Foundation ECS-9873531.

## Footnotes

[1]In this context, "state" is represented by the history suffix. That is, an instance is in a "state" if the instance's incoming history matches the suffix representing the state. In this case, the voting set is exactly the set of instances in the same state as the current choice point $s_t$

## References

[1] Thomas G. Dietterich. The MAXQ method for hierarchical reinforcement learning. In *Autonomous Robots Journal, Special Issue on Learning in Autonomous Robots*, 1998.

[2] Andrew K. McCallum. *Reinforcement Learning with Selective Perception and Hidden State*. PhD thesis, University of Rochester, 1995.

[3] Ron Parr. *Hierarchical Control and Learning for Markov Decision Processes*. PhD thesis, University of California at Berkeley, 1998.

[4] Dana Ron, Yoram Singer, and Naftali Tishby. The power of amnesia: Learning probabilistic automata with variable mem ory length. *Machine Learning*, 25:117–149, 1996.

[5] R. Sutton, D. Precup, and S. Singh. Intra-option learning about temporally abstract actions. In *Proceedings of the 15th International Conference on Machine Learning*, pages 556–564, 1998.
